# Incremental Natural Actor-Critic Algorithms

**Shalabh Bhatnagar**
Department of Computer Science & Automation, Indian Institute of Science, Bangalore, India

**Richard S. Sutton, Mohammad Ghavamzadeh, Mark Lee**
Department of Computing Science, University of Alberta, Edmonton, Alberta, Canada

## Abstract

We present four new reinforcement learning algorithms based on actor-critic and natural-gradient ideas, and provide their convergence proofs. Actor-critic reinforcement learning methods are online approximations to policy iteration in which the value-function parameters are estimated using temporal difference learning and the policy parameters are updated by stochastic gradient descent. Methods based on policy gradients in this way are of special interest because of their compatibility with function approximation methods, which are needed to handle large or infinite state spaces. The use of temporal difference learning in this way is of interest because in many applications it dramatically reduces the variance of the gradient estimates. The use of the natural gradient is of interest because it can produce better conditioned parameterizations and has been shown to further reduce variance in some cases. Our results extend prior two-timescale convergence results for actor-critic methods by Konda and Tsitsiklis by using temporal difference learning in the actor and by incorporating natural gradients, and they extend prior empirical studies of natural actor-critic methods by Peters, Vijayakumar and Schaal by providing the first convergence proofs and the first fully incremental algorithms.

## 1 Introduction

Actor-critic (AC) algorithms are based on the simultaneous online estimation of the parameters of two structures, called the *actor* and the *critic*. The actor corresponds to a conventional action-selection policy, mapping states to actions in a probabilistic manner. The critic corresponds to a conventional value function, mapping states to expected cumulative future reward. Thus, the critic addresses a problem of prediction, whereas the actor is concerned with control. These problems are separable, but are solved simultaneously to find an optimal policy, as in policy iteration. A variety of methods can be used to solve the prediction problem, but the ones that have proved most effective in large applications are those based on some form of temporal difference (TD) learning (Sutton, 1988) in which estimates are updated on the basis of other estimates. Such *bootstrapping* methods can be viewed as a way of accelerating learning by trading bias for variance.

Actor-critic methods were among the earliest to be investigated in reinforcement learning (Barto et al., 1983; Sutton, 1984). They were largely supplanted in the 1990's by methods that estimate action-value functions and use them directly to select actions without an explicit policy structure. This approach was appealing because of its simplicity, but when combined with function approximation was found to have theoretical difficulties including in some cases a failure to converge. These problems led to renewed interest in methods with an explicit representation of the policy, which came to be known as policy gradient methods (Marbach, 1998; Sutton et al., 2000; Konda & Tsitsiklis, 2000; Baxter & Bartlett, 2001). Policy gradient methods without bootstrapping can be easily proved convergent, but converge slowly because of the high variance of their gradient estimates. Combining them with bootstrapping is a promising avenue toward a more effective method.

Another approach to speeding up policy gradient algorithms was proposed by Kakade (2002) and then refined and extended by Bagnell and Schneider (2003) and by Peters et al. (2003). The idea

was to replace the policy gradient with the so-called *natural* policy gradient. This was motivated by the intuition that a change in the policy parameterization should not influence the result of the policy update. In terms of the policy update rule, the move to the natural gradient amounts to linearly transforming the gradient using the inverse Fisher information matrix of the policy.

In this paper, we introduce four new AC algorithms, three of which incorporate natural gradients. All the algorithms are for the average reward setting and use function approximation in the state-value function. For all four methods we prove convergence of the parameters of the policy and state-value function to a local maximum of a performance function that corresponds to the average reward plus a measure of the TD error inherent in the function approximation. Due to space limitations, we do not present the convergence analysis of our algorithms here; it can be found, along with some empirical results using our algorithms, in the extended version of this paper (Bhatnagar et al., 2007). Our results extend prior AC methods, especially those of Konda and Tsitsiklis (2000) and of Peters et al. (2005). We discuss these relationships in detail in Section 6. Our analysis does not cover the use of eligibility traces but we believe the extension to that case would be straightforward.

## 2   The Policy Gradient Framework

We consider the standard reinforcement learning framework (e.g., see Sutton & Barto, 1998), in which a learning agent interacts with a stochastic environment and this interaction is modeled as a discrete-time Markov decision process. The state, action, and reward at each time $t \in \{0, 1, 2, \ldots\}$ are denoted $s_t \in \mathcal{S}$, $a_t \in \mathcal{A}$, and $r_t \in \mathbb{R}$ respectively. We assume the reward is random, real-valued, and uniformly bounded. The environment's dynamics are characterized by state-transition probabilities $p(s'|s,a) = \Pr(s_{t+1} = s'|s_t = s, a_t = a)$, and single-stage expected rewards $r(s,a) = \mathbf{E}[r_{t+1}|s_t = s, a_t = a]$, $\forall s, s' \in \mathcal{S}$, $\forall a \in \mathcal{A}$. The agent selects an action at each time $t$ using a randomized stationary policy $\pi(a|s) = \Pr(a_t = a|s_t = s)$. We assume

**(B1)** *The Markov chain induced by any policy is irreducible and aperiodic.*

The long-term average reward per step under policy $\pi$ is defined as

$$J(\pi) = \lim_{T\to\infty} \frac{1}{T}\mathbf{E}\left[\sum_{t=0}^{T-1} r_{t+1}|\pi\right] = \sum_{s\in\mathcal{S}} d^\pi(s) \sum_{a\in\mathcal{A}} \pi(a|s)r(s,a),$$

where $d^\pi(s)$ is the stationary distribution of state $s$ under policy $\pi$. The limit here is well-defined under (B1). Our aim is to find a policy $\pi^*$ that maximizes the average reward, i.e., $\pi^* = \arg\max_\pi J(\pi)$. In the average reward formulation, a policy $\pi$ is assessed according to the expected differential reward associated with states $s$ or state–action pairs $(s, a)$. For all states $s \in \mathcal{S}$ and actions $a \in \mathcal{A}$, the differential action-value function and the differential state-value function under policy $\pi$ are defined as[1]

$$Q^\pi(s,a) = \sum_{t=0}^{\infty} \mathbf{E}[r_{t+1} - J(\pi)|s_0 = s, a_0 = a, \pi] \quad , \quad V^\pi(s) = \sum_{a\in\mathcal{A}} \pi(a|s)Q^\pi(s,a). \quad (1)$$

In policy gradient methods, we define a class of parameterized stochastic policies $\{\pi(\cdot|s;\boldsymbol{\theta}), s \in \mathcal{S}, \boldsymbol{\theta} \in \Theta\}$, estimate the gradient of the average reward with respect to the policy parameters $\boldsymbol{\theta}$ from the observed states, actions, and rewards, and then improve the policy by adjusting its parameters in the direction of the gradient. Since in this setting a policy $\pi$ is represented by its parameters $\boldsymbol{\theta}$, policy dependent functions such as $J(\pi)$, $d^\pi(\cdot)$, $V^\pi(\cdot)$, and $Q^\pi(\cdot, \cdot)$ can be written as $J(\boldsymbol{\theta})$, $d(\cdot; \boldsymbol{\theta})$, $V(\cdot; \boldsymbol{\theta})$, and $Q(\cdot, \cdot; \boldsymbol{\theta})$, respectively. We assume

**(B2)** *For any state–action pair $(s, a)$, policy $\pi(a|s; \boldsymbol{\theta})$ is continuously differentiable in the parameters $\boldsymbol{\theta}$.*

Previous works (Marbach, 1998; Sutton et al., 2000; Baxter & Bartlett, 2001) have shown that the gradient of the average reward for parameterized policies that satisfy (B1) and (B2) is given by[2]

$$\nabla J(\pi) = \sum_{s\in\mathcal{S}} d^\pi(s) \sum_{a\in\mathcal{A}} \nabla\pi(a|s)Q^\pi(s,a). \quad (2)$$

Observe that if $b(s)$ is any given function of $s$ (also called a *baseline*), then

$$\sum_{s \in \mathcal{S}} d^\pi(s) \sum_{a \in \mathcal{A}} \nabla \pi(a|s) b(s) = \sum_{s \in \mathcal{S}} d^\pi(s) b(s) \nabla \left( \sum_{a \in \mathcal{A}} \pi(a|s) \right) = \sum_{s \in \mathcal{S}} d^\pi(s) b(s) \nabla(1) = 0,$$

and thus, for any baseline $b(s)$, the gradient of the average reward can be written as

$$\nabla J(\pi) = \sum_{s \in \mathcal{S}} d^\pi(s) \sum_{a \in \mathcal{A}} \nabla \pi(a|s)(Q^\pi(s,a) \pm b(s)). \tag{3}$$

The baseline can be chosen such in a way that the variance of the gradient estimates is minimized (Greensmith et al., 2004).

The *natural* gradient, denoted $\tilde{\nabla} J(\pi)$, can be calculated by linearly transforming the regular gradient, using the inverse Fisher information matrix of the policy: $\tilde{\nabla} J(\pi) = \boldsymbol{G}^{-1}(\boldsymbol{\theta}) \nabla J(\pi)$. The Fisher information matrix $\boldsymbol{G}(\boldsymbol{\theta})$ is positive definite and symmetric, and is given by

$$\boldsymbol{G}(\boldsymbol{\theta}) = \mathbf{E}_{s \sim d^\pi, a \sim \pi}[\nabla \log \pi(a|s) \nabla \log \pi(a|s)^\top]. \tag{4}$$

## 3   Policy Gradient with Function Approximation

Now consider the case in which the action-value function for a fixed policy $\pi$, $Q^\pi$, is approximated by a learned function approximator. If the approximation is sufficiently good, we might hope to use it in place of $Q^\pi$ in Eqs. 2 and 3, and still point roughly in the direction of the true gradient. Sutton et al. (2000) showed that if the approximation $\hat{Q}^\pi_{\boldsymbol{w}}$ with parameters $\boldsymbol{w}$ is *compatible*, i.e., $\nabla_{\boldsymbol{w}} \hat{Q}^\pi_{\boldsymbol{w}}(s,a) = \nabla \log \pi(a|s)$, and minimizes the mean squared error

$$\mathcal{E}^\pi(\boldsymbol{w}) = \sum_{s \in \mathcal{S}} d^\pi(s) \sum_{a \in \mathcal{A}} \pi(a|s)[Q^\pi(s,a) - \hat{Q}^\pi_{\boldsymbol{w}}(s,a)]^2 \tag{5}$$

for parameter value $\boldsymbol{w}^*$, then we can replace $Q^\pi$ with $\hat{Q}^\pi_{\boldsymbol{w}^*}$ in Eqs. 2 and 3. Thus, we work with a linear approximation $\hat{Q}^\pi_{\boldsymbol{w}}(s,a) = \boldsymbol{w}^\top \boldsymbol{\psi}(s,a)$, in which the $\boldsymbol{\psi}(s,a)$'s are compatible features defined according to $\boldsymbol{\psi}(s,a) = \nabla \log \pi(a|s)$. Note that compatible features are well defined under (B2). The Fisher information matrix of Eq. 4 can be written using the compatible features as

$$\boldsymbol{G}(\boldsymbol{\theta}) = \mathbf{E}_{s \sim d^\pi, a \sim \pi}[\boldsymbol{\psi}(s,a) \boldsymbol{\psi}(s,a)^\top]. \tag{6}$$

Suppose $\mathcal{E}^\pi(\boldsymbol{w})$ denotes the mean squared error

$$\mathcal{E}^\pi(\boldsymbol{w}) = \sum_{s \in \mathcal{S}} d^\pi(s) \sum_{a \in \mathcal{A}} \pi(a|s)[Q^\pi(s,a) - \boldsymbol{w}^\top \boldsymbol{\psi}(s,a) - b(s)]^2 \tag{7}$$

of our compatible linear parameterized approximation $\boldsymbol{w}^\top \boldsymbol{\psi}(s,a)$ and an arbitrary baseline $b(s)$. Let $\boldsymbol{w}^* = \arg\min_{\boldsymbol{w}} \mathcal{E}^\pi(\boldsymbol{w})$ denote the optimal parameter. Lemma 1 shows that the value of $\boldsymbol{w}^*$ does not depend on the given baseline $b(s)$; as a result the mean squared error problems of Eqs. 5 and 7 have the same solutions. Lemma 2 shows that if the parameter is set to be equal to $\boldsymbol{w}^*$, then the resulting mean squared error $\mathcal{E}^\pi(\boldsymbol{w}^*)$ (now treated as a function of the baseline $b(s)$) is further minimized when $b(s) = V^\pi(s)$. In other words, the variance in the action-value-function estimator is minimized if the baseline is chosen to be the state-value function itself.[3]

**Lemma 1** *The optimum weight parameter $\boldsymbol{w}^*$ for any given $\boldsymbol{\theta}$ (policy $\pi$) satisfies*[4]

$$\boldsymbol{w}^* = \boldsymbol{G}^{-1}(\boldsymbol{\theta}) \mathbf{E}_{s \sim d^\pi, a \sim \pi}[Q^\pi(s,a) \boldsymbol{\psi}(s,a)].$$

**Proof**   Note that

$$\nabla_{\boldsymbol{w}} \mathcal{E}^\pi(\boldsymbol{w}) = -2 \sum_{s \in \mathcal{S}} d^\pi(s) \sum_{a \in \mathcal{A}} \pi(a|s)[Q^\pi(s,a) - w^\top \boldsymbol{\psi}(s,a) - b(s)] \boldsymbol{\psi}(s,a). \tag{8}$$

Equating the above to zero, one obtains

$$\sum_{s \in \mathcal{S}} d^\pi(s) \sum_{a \in \mathcal{A}} \pi(a|s) \boldsymbol{\psi}(s,a) \boldsymbol{\psi}(s,a)^\top \boldsymbol{w}^* = \sum_{s \in \mathcal{S}} d^\pi(s) \sum_{a \in \mathcal{A}} \pi(a|s) Q^\pi(s,a) \boldsymbol{\psi}(s,a) - \sum_{s \in \mathcal{S}} d^\pi(s) \sum_{a \in \mathcal{A}} \pi(a|s) b(s) \boldsymbol{\psi}(s,a).$$

The last term on the right-hand side equals zero because $\sum_{a \in \mathcal{A}} \pi(a|s)\boldsymbol{\psi}(s,a) = \sum_{a \in \mathcal{A}} \nabla \pi(a|s) = 0$ for any state $s$. Now, from Eq. 8, the Hessian $\nabla_{\boldsymbol{w}}^2 \mathcal{E}^\pi(\boldsymbol{w})$ evaluated at $\boldsymbol{w}^*$ can be seen to be $2\boldsymbol{G}(\boldsymbol{\theta})$. The claim follows because $\boldsymbol{G}(\boldsymbol{\theta})$ is positive definite for any $\boldsymbol{\theta}$. $\qquad\square$

Next, given the optimum weight parameter $\boldsymbol{w}^*$, we obtain the minimum variance baseline in the action-value-function estimator corresponding to policy $\pi$. Thus we consider now $\mathcal{E}^\pi(\boldsymbol{w}^*)$ as a function of the baseline $b$, and obtain $b^* = \arg\min_b \mathcal{E}^\pi(\boldsymbol{w}^*)$.

**Lemma 2** *For any given policy $\pi$, the minimum variance baseline $b^*(s)$ in the action-value-function estimator corresponds to the state-value function $V^\pi(s)$.*

**Proof** For any $s \in \mathcal{S}$, let $\mathcal{E}^{\pi,s}(\boldsymbol{w}^*) = \sum_{a \in \mathcal{A}} \pi(a|s)[Q^\pi(s,a) - \boldsymbol{w}^{*\top}\boldsymbol{\psi}(s,a) - b(s)]^2$. Then $\mathcal{E}^\pi(\boldsymbol{w}^*) = \sum_{s \in \mathcal{S}} d^\pi(s)\mathcal{E}^{\pi,s}(\boldsymbol{w}^*)$. Note that by (B1), the Markov chain corresponding to any policy $\pi$ is positive recurrent because the number of states is finite. Hence, $d^\pi(s) > 0$ for all $s \in \mathcal{S}$. Thus, one needs to find the baseline $b(s)$ that minimizes $\mathcal{E}^{\pi,s}(\boldsymbol{w}^*)$ for each $s \in \mathcal{S}$. For any $s \in \mathcal{S}$,

$$\frac{\partial \mathcal{E}^{\pi,s}(\boldsymbol{w}^*)}{\partial b(s)} = -2 \sum_{a \in \mathcal{A}} \pi(a|s)[Q^\pi(s,a) - \boldsymbol{w}^{*\top}\boldsymbol{\psi}(s,a) - b(s)].$$

Equating the above to zero, we obtain

$$b^*(s) = \sum_{a \in \mathcal{A}} \pi(a|s)Q^\pi(s,a) - \sum_{a \in \mathcal{A}} \pi(a|s)\boldsymbol{w}^{*\top}\boldsymbol{\psi}(s,a).$$

The rightmost term equals zero because $\sum_{a \in \mathcal{A}} \pi(a|s)\boldsymbol{\psi}(s,a) = 0$. Hence $b^*(s) = \sum_{a \in \mathcal{A}} \pi(a|s) Q^\pi(s,a) = V^\pi(s)$. The second derivative of $\mathcal{E}^{\pi,s}(\boldsymbol{w}^*)$ w.r.t. $b(s)$ equals 2. The claim follows. $\quad\square$

From Lemmas 1 and 2, $\boldsymbol{w}^{*\top}\boldsymbol{\psi}(s,a)$ is a least-squared optimal parametric representation for the *advantage* function $A^\pi(s,a) = Q^\pi(s,a) - V^\pi(s)$ as well as for the action-value function $Q^\pi(s,a)$. However, because $\mathbf{E}_{a \sim \pi}[\boldsymbol{w}^\top\boldsymbol{\psi}(s,a)] = \sum_{a \in \mathcal{A}} \pi(a|s)\boldsymbol{w}^\top\boldsymbol{\psi}(s,a) = 0$, $\forall s \in \mathcal{S}$, it is better to think of $\boldsymbol{w}^\top\boldsymbol{\psi}(s,a)$ as an approximation of the advantage function rather than of the action-value function.

The TD error $\delta_t$ is a random quantity that is defined according to $\delta_t = r_{t+1} - \hat{J}_{t+1} + \hat{V}(s_{t+1}) - \hat{V}(s_t)$, where $\hat{V}$ and $\hat{J}$ are consistent estimates of the state-value function and the average reward, respectively. Thus, these estimates satisfy $\mathbf{E}[\hat{V}(s_t)|s_t, \pi] = V^\pi(s_t)$ and $\mathbf{E}[\hat{J}_{t+1}|s_t, \pi] = J(\pi)$, for any $t \geq 0$. The next lemma shows that $\delta_t$ is a consistent estimate of the advantage function $A^\pi$.

**Lemma 3** *Under given policy $\pi$, we have $\mathbf{E}[\delta_t|s_t, a_t, \pi] = A^\pi(s_t, a_t)$.*

**Proof** Note that

$\mathbf{E}[\delta_t|s_t, a_t, \pi] = \mathbf{E}[r_{t+1} - \hat{J}_{t+1} + \hat{V}(s_{t+1}) - \hat{V}(s_t)|s_t, a_t, \pi] = r(s_t, a_t) - J(\pi) + \mathbf{E}[\hat{V}(s_{t+1})|s_t, a_t, \pi] - V^\pi(s_t).$

Now

$\mathbf{E}[\hat{V}(s_{t+1})|s_t, a_t, \pi] = \mathbf{E}[\mathbf{E}[\hat{V}(s_{t+1})|s_{t+1}, \pi]|s_t, a_t, \pi] = \mathbf{E}[V^\pi(s_{t+1})|s_t, a_t] = \sum_{s_{t+1} \in \mathcal{S}} p(s_{t+1}|s_t, a_t)V^\pi(s_{t+1}).$

Also $r(s_t, a_t) - J(\pi) + \sum_{s_{t+1} \in \mathcal{S}} p(s_{t+1}|s_t, a_t)V^\pi(s_{t+1}) = Q^\pi(s_t, a_t)$. The claim follows. $\quad\square$

By setting the baseline $b(s)$ equal to the value function $V^\pi(s)$, Eq. 3 can be written as $\nabla J(\pi) = \sum_{s \in \mathcal{S}} d^\pi(s) \sum_{a \in \mathcal{A}} \pi(a|s)\boldsymbol{\psi}(s,a)A^\pi(s,a)$. From Lemma 3, $\delta_t$ is a consistent estimate of the advantage function $A^\pi(s,a)$. Thus, $\widehat{\nabla J}(\pi) = \delta_t\boldsymbol{\psi}(s_t, a_t)$ is a consistent estimate of $\nabla J(\pi)$. However, calculating $\delta_t$ requires having estimates, $\hat{J}$, $\hat{V}$, of the average reward and the value function. While an average reward estimate is simple enough to obtain given the single-stage reward function, the same is not necessarily true for the value function. We use function approximation for the value function as well. Suppose $\boldsymbol{f}(s)$ is a feature vector for state $s$. One may then approximate $V^\pi(s)$ with $\boldsymbol{v}^\top\boldsymbol{f}(s)$, where $\boldsymbol{v}$ is a parameter vector that can be tuned (for a fixed policy $\pi$) using a TD algorithm. In our algorithms, we use $\delta_t = r_{t+1} - \hat{J}_{t+1} + \boldsymbol{v}_t^\top\boldsymbol{f}(s_{t+1}) - \boldsymbol{v}_t^\top\boldsymbol{f}(s_t)$ as an estimate for the TD error, where $\boldsymbol{v}_t$ corresponds to the value function parameter at time $t$.

Let $\bar{V}^{\pi}(s) = \sum_{a \in \mathcal{A}} \pi(a|s)[r(s,a) - J(\pi) + \sum_{s' \in \mathcal{S}} p(s'|s,a) \boldsymbol{v}^{\pi\top} \boldsymbol{f}(s')]$, where $\boldsymbol{v}^{\pi\top} \boldsymbol{f}(s')$ is an estimate of the value function $V^{\pi}(s')$ that is obtained upon convergence viz., $\lim_{t \to \infty} \boldsymbol{v}_t = \boldsymbol{v}^{\pi}$ with probability one. Also, let $\delta_t^{\pi} = r_{t+1} - \hat{J}_{t+1} + \boldsymbol{v}^{\pi\top} \boldsymbol{f}(s_{t+1}) - \boldsymbol{v}^{\pi\top} \boldsymbol{f}(s_t)$, where $\delta_t^{\pi}$ corresponds to a stationary estimate of the TD error with function approximation under policy $\pi$.

**Lemma 4** $\mathbf{E}[\delta_t^{\pi} \boldsymbol{\psi}(s_t, a_t)|\boldsymbol{\theta}] = \nabla J(\pi) + \sum_{s \in \mathcal{S}} d^{\pi}(s)[\nabla \bar{V}^{\pi}(s) - \nabla \boldsymbol{v}^{\pi\top} \boldsymbol{f}(s)]$.

Proof of this lemma can be found in the extended version of this paper (Bhatnagar et al., 2007). Note that $\mathbf{E}[\delta_t \boldsymbol{\psi}(s_t, a_t)|\boldsymbol{\theta}] = \nabla J(\pi)$, provided $\delta_t$ is defined as $\delta_t = r_{t+1} - \hat{J}_{t+1} + \hat{V}(s_{t+1}) - \hat{V}(s_t)$ (as was considered in Lemma 3). For the case with function approximation that we study, from Lemma 4, the quantity $\sum_{s \in \mathcal{S}} d^{\pi}(s)[\nabla \bar{V}^{\pi}(s) - \nabla \boldsymbol{v}^{\pi\top} \boldsymbol{f}(s)]$ may be viewed as the error or bias in the estimate of the gradient of average reward that results from the use of function approximation.

## 4 Actor-Critic Algorithms

We present four new AC algorithms in this section. These algorithms are in the general form shown in Table 1. They update the policy parameters along the direction of the average-reward gradient. While estimates of the regular gradient are used for this purpose in Algorithm 1, natural gradient estimates are used in Algorithms 2–4. While critic updates in our algorithms can be easily extended to the case of TD($\lambda$), $\lambda > 0$, we restrict our attention to the case when $\lambda = 0$. In addition to assumptions (B1) and (B2), we make the following assumption:

**(B3)** *The step-size schedules for the critic $\{\alpha_t\}$ and the actor $\{\beta_t\}$ satisfy*

$$\sum_t \alpha_t = \sum_t \beta_t = \infty \quad , \quad \sum_t \alpha_t^2 , \sum_t \beta_t^2 < \infty \quad , \quad \lim_{t \to \infty} \frac{\beta_t}{\alpha_t} = 0. \tag{9}$$

As a consequence of Eq. 9, $\beta_t \to 0$ faster than $\alpha_t$. Hence the critic has uniformly higher increments than the actor beyond some $t_0$, and thus it converges faster than the actor.

Table 1: A Template for Incremental AC Algorithms.

| | | |
|---|---|---|
| 1: | **Input:** | |
| | • Randomized parameterized policy $\pi(\cdot|\cdot; \boldsymbol{\theta})$, | |
| | • Value function feature vector $\boldsymbol{f}(s)$. | |
| 2: | **Initialization:** | |
| | • Policy parameters $\boldsymbol{\theta} = \boldsymbol{\theta}_0$, | |
| | • Value function weight vector $\boldsymbol{v} = \boldsymbol{v}_0$, | |
| | • Step sizes $\alpha = \alpha_0, \ \beta = \beta_0, \ \xi = c\alpha_0$, | |
| | • Initial state $s_0$. | |
| 3: | **for** $t = 0, 1, 2, \ldots$ **do** | |
| 4: | **Execution:** | |
| | • Draw action $a_t \sim \pi(a_t|s_t; \boldsymbol{\theta}_t)$, | |
| | • Observe next state $s_{t+1} \sim p(s_{t+1}|s_t, a_t)$, | |
| | • Observe reward $r_{t+1}$. | |
| 5: | **Average Reward Update:** | $\hat{J}_{t+1} = (1 - \xi_t)\hat{J}_t + \xi_t r_{t+1}$ |
| 6: | **TD error:** | $\delta_t = r_{t+1} - \hat{J}_{t+1} + \boldsymbol{v}_t^\top \boldsymbol{f}(s_{t+1}) - \boldsymbol{v}_t^\top \boldsymbol{f}(s_t)$ |
| 7: | **Critic Update:** | algorithm specific (see the text) |
| 8: | **Actor Update:** | algorithm specific (see the text) |
| 9: | **endfor** | |
| 10: | **return** Policy and value-function parameters $\boldsymbol{\theta}, \boldsymbol{v}$ | |

We now present the critic and the actor updates of our four AC algorithms.

**Algorithm 1 (Regular-Gradient AC):**

| | |
|---|---|
| **Critic Update:** | $\boldsymbol{v}_{t+1} = \boldsymbol{v}_t + \alpha_t \delta_t \boldsymbol{f}(s_t)$, |
| **Actor Update:** | $\boldsymbol{\theta}_{t+1} = \boldsymbol{\theta}_t + \beta_t \delta_t \boldsymbol{\psi}(s_t, a_t)$. |

This is the only AC algorithm presented in the paper that is based on the regular gradient estimate. This algorithm stores two parameter vectors $\boldsymbol{\theta}$ and $\boldsymbol{v}$. Its per time-step computational cost is linear in the number of policy and value-function parameters.

The next algorithm is based on the natural-gradient estimate $\tilde{\nabla}J(\boldsymbol{\theta}_t) = \boldsymbol{G}^{-1}(\boldsymbol{\theta}_t)\delta_t\boldsymbol{\psi}(s_t, a_t)$ in place of the regular-gradient estimate in Algorithm 1. We derive a procedure for recursively estimating $\boldsymbol{G}^{-1}(\boldsymbol{\theta})$ and show in Lemma 5 that our estimate $\boldsymbol{G}_t^{-1}$ converges to $\boldsymbol{G}^{-1}(\boldsymbol{\theta})$ as $t \rightarrow \infty$ with probability one. This is required for proving convergence of this algorithm. The Fisher information matrix can be estimated in an online manner as $\boldsymbol{G}_{t+1} = \frac{1}{t+1}\sum_{i=0}^{t}\boldsymbol{\psi}(s_i, a_i)\boldsymbol{\psi}^{\top}(s_i, a_i)$. One may obtain recursively $\boldsymbol{G}_{t+1} = (1 - \frac{1}{t+1})\boldsymbol{G}_t + \frac{1}{t+1}\boldsymbol{\psi}(s_t, a_t)\boldsymbol{\psi}^{\top}(s_t, a_t)$, or more generally

$$\boldsymbol{G}_{t+1} = (1 - \zeta_t)\boldsymbol{G}_t + \zeta_t\boldsymbol{\psi}(s_t, a_t)\boldsymbol{\psi}^{\top}(s_t, a_t). \tag{10}$$

Using the Sherman-Morrison matrix inversion lemma, one obtains

$$\boldsymbol{G}_{t+1}^{-1} = \frac{1}{1 - \zeta_t}\left[\boldsymbol{G}_t^{-1} - \zeta_t\frac{\boldsymbol{G}_t^{-1}\boldsymbol{\psi}(s_t, a_t)(\boldsymbol{G}_t^{-1}\boldsymbol{\psi}(s_t, a_t))^{\top}}{1 - \zeta_t + \zeta_t\boldsymbol{\psi}(s_t, a_t)^{\top}\boldsymbol{G}_t^{-1}\boldsymbol{\psi}(s_t, a_t)}\right] \tag{11}$$

For our Alg. 2 and 4, we require the following additional assumption for the convergence analysis:

**(B4)** *The iterates $\boldsymbol{G}_t$ and $\boldsymbol{G}_t^{-1}$ satisfy $\sup_{t,\boldsymbol{\theta},s,a} \| \boldsymbol{G}_t \|$ and $\sup_{t,\boldsymbol{\theta},s,a} \| \boldsymbol{G}_t^{-1} \| < \infty$.*

**Lemma 5** *For any given parameter $\boldsymbol{\theta}$, $\boldsymbol{G}_t^{-1}$ in Eq. 11 satisfies $\boldsymbol{G}_t^{-1} \rightarrow \boldsymbol{G}^{-1}(\boldsymbol{\theta})$ as $t \rightarrow \infty$ with probability one.*

**Proof** It is easy to see from Eq. 10 that $\boldsymbol{G}_t \rightarrow \boldsymbol{G}(\boldsymbol{\theta})$ as $t \rightarrow \infty$ with probability one, for any given $\boldsymbol{\theta}$ held fixed. For a fixed $\boldsymbol{\theta}$,

$$\| \boldsymbol{G}_t^{-1} - \boldsymbol{G}^{-1}(\boldsymbol{\theta}) \| = \| \boldsymbol{G}^{-1}(\boldsymbol{\theta})(\boldsymbol{G}(\boldsymbol{\theta})\boldsymbol{G}_t^{-1} - \boldsymbol{I}) \| = \| \boldsymbol{G}^{-1}(\boldsymbol{\theta})(\boldsymbol{G}(\boldsymbol{\theta}) - \boldsymbol{G}_t)\boldsymbol{G}_t^{-1} \| \le$$
$$\sup_{\boldsymbol{\theta}} \| \boldsymbol{G}^{-1}(\boldsymbol{\theta}) \| \sup_{t,\boldsymbol{\theta},s,a} \| \boldsymbol{G}_t^{-1} \| \cdot \| \boldsymbol{G}(\boldsymbol{\theta}) - \boldsymbol{G}_t \| \rightarrow 0 \quad \text{as} \quad t \rightarrow \infty$$

by assumption (B4). The claim follows. $\qquad\square$

Our second algorithm stores a matrix $\boldsymbol{G}^{-1}$ and two parameter vectors $\boldsymbol{\theta}$ and $\boldsymbol{v}$. Its per time-step computational cost is linear in the number of value-function parameters and quadratic in the number of policy parameters.

**Algorithm 2 (Natural-Gradient AC with Fisher Information Matrix):**

$$\text{Critic Update:} \qquad \boldsymbol{v}_{t+1} = \boldsymbol{v}_t + \alpha_t\delta_t\boldsymbol{f}(s_t),$$
$$\text{Actor Update:} \qquad \boldsymbol{\theta}_{t+1} = \boldsymbol{\theta}_t + \beta_t\boldsymbol{G}_{t+1}^{-1}\delta_t\boldsymbol{\psi}(s_t, a_t),$$

with the estimate of the inverse Fisher information matrix updated according to Eq. 11. We let $\boldsymbol{G}_0^{-1} = k\boldsymbol{I}$, where $k$ is a positive constant. Thus $\boldsymbol{G}_0^{-1}$ and $\boldsymbol{G}_0$ are positive definite and symmetric matrices. From Eq. 10, $\boldsymbol{G}_t$, $t > 0$ can be seen to be positive definite and symmetric because these are convex combinations of positive definite and symmetric matrices. Hence, $\boldsymbol{G}_t^{-1}$, $t > 0$, are positive definite and symmetric as well.

As mentioned in Section 3, it is better to think of the compatible approximation $\boldsymbol{w}^{\top}\boldsymbol{\psi}(s, a)$ as an approximation of the advantage function rather than of the action-value function. In our next algorithm we tune the parameters $\boldsymbol{w}$ in such a way as to minimize an estimate of the least-squared error $\mathcal{E}^{\pi}(\boldsymbol{w}) = \mathbf{E}_{s \sim d^{\pi}, a \sim \pi}[(\boldsymbol{w}^{\top}\boldsymbol{\psi}(s, a) - A^{\pi}(s, a))^2]$. The gradient of $\mathcal{E}^{\pi}(\boldsymbol{w})$ is thus $\nabla_{\boldsymbol{w}}\mathcal{E}^{\pi}(\boldsymbol{w}) = 2\mathbf{E}_{s \sim d^{\pi}, a \sim \pi}[(\boldsymbol{w}^{\top}\boldsymbol{\psi}(s, a) - A^{\pi}(s, a))\boldsymbol{\psi}(s, a)]$, which can be estimated as $\widehat{\nabla_{\boldsymbol{w}}\mathcal{E}^{\pi}}(\boldsymbol{w}) = 2[\boldsymbol{\psi}(s_t, a_t)\boldsymbol{\psi}(s_t, a_t)^{\top}\boldsymbol{w} - \delta_t\boldsymbol{\psi}(s_t, a_t)]$. Hence, we update advantage parameters $\boldsymbol{w}$ along with value-function parameters $\boldsymbol{v}$ in the critic update of this algorithm. As with Peters et al. (2005), we use the natural gradient estimate $\tilde{\nabla}J(\boldsymbol{\theta}_t) = \boldsymbol{w}_{t+1}$ in the actor update of Alg. 3. This algorithm stores three parameter vectors, $\boldsymbol{v}$, $\boldsymbol{w}$, and $\boldsymbol{\theta}$. Its per time-step computational cost is linear in the number of value-function parameters and quadratic in the number of policy parameters.

**Algorithm 3** **(Natural-Gradient AC with Advantage Parameters):**

$$\text{Critic Update:} \quad \boldsymbol{v}_{t+1} = \boldsymbol{v}_t + \alpha_t \delta_t \boldsymbol{f}(s_t),$$
$$\boldsymbol{w}_{t+1} = [\boldsymbol{I} - \alpha_t \boldsymbol{\psi}(s_t, a_t) \boldsymbol{\psi}(s_t, a_t)^\top] \boldsymbol{w}_t + \alpha_t \delta_t \boldsymbol{\psi}(s_t, a_t),$$
$$\text{Actor Update:} \quad \boldsymbol{\theta}_{t+1} = \boldsymbol{\theta}_t + \beta_t \boldsymbol{w}_{t+1}.$$

Although an estimate of $\boldsymbol{G}^{-1}(\boldsymbol{\theta})$ is not explicitly computed and used in Algorithm 3, the convergence analysis of this algorithm shows that the overall scheme still moves in the direction of the natural gradient of average reward. In Algorithm 4, however, we explicitly estimate $\boldsymbol{G}^{-1}(\boldsymbol{\theta})$ (as in Algorithm 2), and use it in the critic update for $\boldsymbol{w}$. The overall scheme is again seen to follow the direction of the natural gradient of average reward. Here, we let $\tilde{\nabla}_{\boldsymbol{w}} \mathcal{E}^\pi(\boldsymbol{w}) = 2 \boldsymbol{G}_t^{-1} [\boldsymbol{\psi}(s_t, a_t) \boldsymbol{\psi}(s_t, a_t)^\top \boldsymbol{w} - \delta_t \boldsymbol{\psi}(s_t, a_t)]$ be the estimate of the natural gradient of the least-squared error $\mathcal{E}^\pi(\boldsymbol{w})$. This also simplifies the critic update for $\boldsymbol{w}$. Algorithm 4 stores a matrix $\boldsymbol{G}^{-1}$ and three parameter vectors, $\boldsymbol{v}$, $\boldsymbol{w}$, and $\boldsymbol{\theta}$. Its per time-step computational cost is linear in the number of value-function parameters and quadratic in the number of policy parameters.

**Algorithm 4** **(Natural-Gradient AC with Advantage Parameters and Fisher Information Matrix):**

$$\text{Critic Update:} \quad \boldsymbol{v}_{t+1} = \boldsymbol{v}_t + \alpha_t \delta_t \boldsymbol{f}(s_t),$$
$$\boldsymbol{w}_{t+1} = (1 - \alpha_t) \boldsymbol{w}_t + \alpha_t \boldsymbol{G}_{t+1}^{-1} \delta_t \boldsymbol{\psi}(s_t, a_t),$$
$$\text{Actor Update:} \quad \boldsymbol{\theta}_{t+1} = \boldsymbol{\theta}_t + \beta_t \boldsymbol{w}_{t+1},$$

where the estimate of the inverse Fisher information matrix is updated according to Eq. 11.

## 5 Convergence of Our Actor-Critic Algorithms

Since our algorithms are gradient-based, one cannot expect to prove convergence to a globally optimal policy. The best that one could hope for is convergence to a local maximum of $J(\boldsymbol{\theta})$. However, because the critic will generally converge to an approximation of the desired projection of the value function (defined by the value function features $\boldsymbol{f}$) in these algorithms, the corresponding convergence results are necessarily weaker, as indicated by the following theorem.

**Theorem 1** *For the parameter iterations in Algorithms 1-4,[5] we have $(\hat{J}_t, \boldsymbol{v}_t, \boldsymbol{\theta}_t) \rightarrow \{(J(\boldsymbol{\theta}^*), \boldsymbol{v}^{\boldsymbol{\theta}^*}, \boldsymbol{\theta}^*) | \boldsymbol{\theta}^* \in \mathcal{Z}\}$ as $t \rightarrow \infty$ with probability one, where the set $\mathcal{Z}$ corresponds to the set of local maxima of a performance function whose gradient is $\mathbf{E}[\delta_t^\pi \boldsymbol{\psi}(s_t, a_t) | \boldsymbol{\theta}]$ (cf. Lemma 4).*

For the proof of this theorem, please refer to Section 6 (Convergence Analysis) of the extended version of this paper (Bhatnagar et al., 2007). This theorem indicates that the policy and state-value-function parameters converge to a local maximum of a performance function that corresponds to the average reward plus a measure of the TD error inherent in the function approximation.

## 6 Relation to Previous Algorithms

**Actor-Critic Algorithm of Konda and Tsitsiklis (2000)**: Unlike our Alg. 2–4, their algorithm does not use estimates of the natural gradient in its actor's update. Their algorithm is similar to our Alg. 1, but with some key differences. **1)** Konda's algorithm uses the Markov process of state–action pairs, and thus its critic update is based on an action-value function. Alg. 1 uses the state process, and therefore its critic update is based on a state-value function. **2)** Whereas Alg. 1 uses a TD error in both critic and actor recursions, Konda's algorithm uses a TD error only in its critic update. The actor recursion in Konda's algorithm uses an action-value estimate instead. Because the TD error is a consistent estimate of the advantage function (Lemma 3), the actor recursion in Alg. 1 uses estimates of advantages instead of action-values, which may result in lower variances. **3)** The convergence analysis of Konda's algorithm is based on the martingale approach and aims at bounding error terms and directly showing convergence; convergence to a local optimum is shown when a TD(1) critic is used. For the case where $\lambda < 1$, they show that given an $\epsilon > 0$, there exists $\lambda$ close enough to one such that when a TD($\lambda$) critic is used, one gets $\liminf_t |\nabla J(\boldsymbol{\theta}_t)| < \epsilon$ with

probability one. Unlike Konda and Tsitsiklis, we primarily use the ordinary differential equation (ODE) based approach for our convergence analysis. Though we use martingale arguments in our analysis, these are restricted to showing that the noise terms asymptotically diminish; the resulting scheme can be viewed as an Euler-discretization of the associated ODE.

**Natural Actor-Critic Algorithm of Peters et al. (2005)**: Our Algorithms 2–4 extend their algorithm by being fully incremental and in that we provide convergence proofs. Peters's algorithm uses a least-squares TD method in its critic's update, whereas all our algorithms are fully incremental. It is not clear how to satisfactorily incorporate least-squares TD methods in a context in which the policy is changing, and our proof techniques do not immediately extend to this case.

## 7   Conclusions and Future Work

We have introduced and analyzed four AC algorithms utilizing both linear function approximation and bootstrapping, a combination which seems essential to large-scale applications of reinforcement learning. All of the algorithms are based on existing ideas such as TD-learning, natural policy gradients, and two-timescale stochastic approximation, but combined in new ways. The main contribution of this paper is proving convergence of the algorithms to a local maximum in the space of policy and value-function parameters. Our Alg. 2–4 are explorations of the use of natural gradients within an AC architecture. The way we use natural gradients is distinctive in that it is totally incremental: the policy is changed on every time step, yet the gradient computation is never reset as it is in the algorithm of Peters et al. (2005). Alg. 3 is perhaps the most interesting of the three natural-gradient algorithms. It never explicitly stores an estimate of the inverse Fisher information matrix and, as a result, it requires less computation. In empirical experiments using our algorithms (not reported here) we observed that it is easier to find good parameter settings for Alg. 3 than it is for the other natural-gradient algorithms and, perhaps because of this, it converged more rapidly than the others and than Konda's algorithm. All our algorithms performed better than Konda's algorithm.

There are a number of ways in which our results are limited and suggest future work. **1)** It is important to characterize the quality of the converged solutions, either by bounding the performance loss due to bootstrapping and approximation error, or through a thorough empirical study. **2)** The algorithms can be extended to incorporate eligibility traces and least-squares methods. As discussed earlier, the former seems straightforward whereas the latter requires more fundamental extensions. **3)** Application of the algorithms to real-world problems is needed to assess their ultimate utility.

## Footnotes

[1]From now on in the paper, we use the terms state-value function and action-value function instead of differential state-value function and differential action-value function.

[2]Throughout the paper, we use notation $\nabla$ to denote $\nabla_\theta$ – the gradient w.r.t. the policy parameters.

[3]It is important to note that Lemma 2 is not about the minimum variance baseline for gradient estimation. It is about the minimum variance baseline of the action-value-function estimator.

[4]This lemma is similar to Kakade's (2002) Theorem 1.

[5]The proof of this theorem requires another assumption viz., (A3) in the extended version of this paper (Bhatnagar et al., 2007), in addition to (B1)-(B3) (resp. (B1)-(B4)) for Algorithm 1 and 3 (resp. for Algorithm 2 and 4). This was not included in this paper due to space limitations.

## References

Bagnell, J., & Schneider, J. (2003). Covariant policy search. *Proceedings of the Eighteenth International Joint Conference on Artificial Intelligence*.

Barto, A. G., Sutton, R. S., & Anderson, C. (1983). Neuron-like elements that can solve difficult learning control problems. *IEEE Transaction on Systems, Man and Cybernetics*, *13*, 835–846.

Baxter, J., & Bartlett, P. (2001). Infinite-horizon policy-gradient estimation. *JAIR*, *15*, 319–350.

Bhatnagar, S., Sutton, R. S., Ghavamzadeh, M., & Lee, M. (2007). Natural actor-critic algorithms. *Submitted to Automatica*.

Greensmith, E., Bartlett, P., & Baxter, J. (2004). Variance reduction techniques for gradient estimates in reinforcement learning. *Journal of Machine Learning Research*, *5*, 1471–1530.

Kakade, S. (2002). A natural policy gradient. *Proceedings of NIPS 14*.

Konda, V., & Tsitsiklis, J. (2000). Actor-critic algorithms. *Proceedings of NIPS 12* (pp. 1008–1014).

Marbach, P. (1998). *Simulated-based methods for Markov decision processes*. Doctoral dissertation, MIT.

Peters, J., Vijayakumar, S., & Schaal, S. (2003). Reinforcement learning for humanoid robotics. *Proceedings of the Third IEEE-RAS International Conference on Humanoid Robots*.

Peters, J., Vijayakumar, S., & Schaal, S. (2005). Natural actor-critic. *Proceedings of the Sixteenth European Conference on Machine Learning* (pp. 280–291).

Sutton, R. S. (1984). *Temporal credit assignment in reinforcement learning*. Doctoral dissertation, UMass Amherst.

Sutton, R. S. (1988). Learning to predict by the methods of temporal differences. *Machine Learning*, *3*, 9–44.

Sutton, R. S., & Barto, A. G. (1998). *Reinforcement learning: An introduction*. MIT Press.

Sutton, R. S., McAllester, D., Singh, S., & Mansour, Y. (2000). Policy gradient methods for reinforcement learning with function approximation. *Proceedings of NIPS 12* (pp. 1057–1063).

